# Two Approaches to Optimal Annealing

**Todd K. Leen**
Dept of Comp. Sci. & Engineering
Oregon Graduate Institute of
Science and Technology
P.O.Box 91000, Portland,
Oregon 97291-1000
tleen@cse.ogi.edu

**Bernhard Schottky and David Saad**
Neural Computing Research Group
Dept of Comp. Sci. & Appl. Math.
Aston University
Birmingham, B4 7ET, UK
schottba{saadd}@aston.ac.uk

## Abstract

We employ both master equation and order parameter approaches to analyze the asymptotic dynamics of on-line learning with different learning rate annealing schedules. We examine the relations between the results obtained by the two approaches and obtain new results on the optimal decay coefficients and their dependence on the number of hidden nodes in a two layer architecture.

## 1 Introduction

The asymptotic dynamics of stochastic on-line learning and it's dependence on the annealing schedule adopted for the learning coefficients have been studied for some time in the stochastic approximation literature [1, 2] and more recently in the neural network literature [3, 4, 5]. The latter studies are based on examining the Kramers-Moyal expansion of the master equation for the weight space probability densities. A different approach, based on the deterministic dynamics of macroscopic quantities called order parameters, has been recently presented [6, 7]. This approach enables one to monitor the evolution of the order parameters and the system performance at all times.

In this paper we examine the relation between the two approaches and contrast the results obtained for different learning rate annealing schedules in the asymptotic regime. We employ the order parameter approach to examine the dependence of the dynamics on the number of hidden nodes in a multilayer system. In addition, we report some lesser-known results on non-standard annealing schedules

## 2   Master Equation

Most on-line learning algorithms assume the form $w_{t+1} = w_t + \eta_0/t^p \, H(w_t, x_t)$ where $w_t$ is the weight at time $t$, $x_t$ is the training example, and $H(w,x)$ is the weight update. The description of the algorithm's dynamics in terms of weight space probability densities starts from the master equation

$$P(w', t+1) = \int dw \, \left\langle \delta \left( w' - w - \frac{\eta_0}{t^p} H(w,x) \right) \right\rangle_x P(w,t) \qquad (1)$$

where $\langle \ldots \rangle_x$ indicates averaging with respect to the measure on $x$, $P(w,t)$ is the probability density on weights at time $t$, and $\delta(\ldots)$ is the Dirac function. One may use the Kramers-Moyal expansion of Eq.(1) to derive a partial differential equation for the weight probability density (here in one dimension for simplicity) [3, 4]

$$\partial_t P(w,t) = \sum_{i=1}^{\infty} \frac{(-1)^i}{i!} \left( \frac{\eta_0}{t^p} \right)^i \partial_w^i \left[ \langle H^i(w,x) \rangle_x P(w,t) \right] \, . \qquad (2)$$

Following [3], we make a small noise expansion for (2) by decomposing the weight trajectory into a deterministic and stochastic pieces

$$w \equiv \phi(t) + \left( \frac{\eta_0}{t^p} \right)^{\gamma} \xi \qquad \text{or} \qquad \xi = \left( \frac{\eta_0}{t^p} \right)^{-\gamma} (w - \phi(t)) \qquad (3)$$

where $\phi(t)$ is the deterministic trajectory, and $\xi$ are the fluctuations. Apart from the factor $(\eta_0/t^p)^{\gamma}$ that scales the fluctuations, this is identical to the formulation for constant learning in [3]. The proper value for the unspecified exponent $\gamma$ will emerge from homogeneity requirements. Next, the dependence of the jump moments $\langle H^i(w,x) \rangle_x$ on $\eta_0$ is explicated by a Taylor series expansion about the deterministic path $\phi$. The coefficients in this series expansion are denoted

$$\alpha_i^{(j)} \equiv \partial^j \langle H^i(w,x) \rangle_x / \partial w^j \big|_{w=\phi}$$

Finally one rewrites (2) in terms of $\phi$ and $\xi$ and the expansion of the jump moments, taking care to transform the differential operators in accordance with (3).

These transformations leave equations of motion for $\phi$ and the density $\Pi(\xi, t)$ on the fluctuations

$$\frac{d\phi}{dt} = \left( \frac{\eta_0}{t^p} \right) \alpha_1^{(0)}(\phi) = \left( \frac{\eta_0}{t^p} \right) \langle H(\phi, x) \rangle_x \qquad (4)$$

$$\partial_t \Pi = -\frac{\gamma p}{t} \partial_\xi (\xi \Pi) + \sum_{m=2}^{\infty} \sum_{i=1}^{m} \frac{(-1)^i}{i!(m-i)!} \alpha_i^{(m-i)} \left( \frac{\eta_0}{t^p} \right)^{i(1-2\gamma)+m\gamma} \partial_\xi^i (\xi^{(m-i)} \Pi) \, . \qquad (5)$$

For stochastic descent $H(w,x) = -\nabla_w E(w,x)$ and (4) describes the evolution of $\phi$ as descent on the average cost. The fluctuation equation (5) requires further manipulation whose form depends on the context. For the usual case of descent in a quadratic minimum ($\alpha_1^{(1)} = -G$, minus the cost function curvature), we take $\gamma = 1/2$ to insure that for any $m$, terms in the sum are homogeneous in $\eta_0/t^p$

For *constant* learning rate ($p = 0$), rescaling time as $t \to \eta_0 t$ allows (5) to be written in a form convenient for perturbative analysis in $\eta_0$ Typically, the limit $\eta_0 \to 0$ is invoked and only the lowest order terms in $\eta_0$ retained (e.g. [3]). These comprise a diffusion operator, which results in a Gaussian approximation for equilibrium densities. Higher order terms have been successfully used to calculate corrections to the equilibrium moments in powers of $\eta_0$ [8].

Of primary interest here is the case of annealed learning, as required for convergence of the parameter estimates. Again assuming a quadratic bowl and $\gamma = 1/2$, the first few terms of (5) are

$$\partial_t \Pi = -\frac{p}{2t} \partial_\xi (\xi \Pi) - \alpha_1^{(1)} \frac{\eta_0}{t^p} \partial_\xi (\xi \Pi) + \frac{1}{2} \alpha_2^{(0)} \frac{\eta_0}{t^p} \partial_\xi^2 \Pi + \mathcal{O}\left(\frac{\eta_0}{t^p}\right)^{3/2} . \quad (6)$$

As $t \to \infty$ the right hand side of (6) is dominated by the first three terms (since $0 < p \leq 1$). Precisely which terms dominate depends on $p$.

We will first review the classical case $p = 1$. Asymptotically $\phi \to w^*$, a local optimum. The first three leading terms on the right hand side of (6) are all of order $1/t$. For $t \to \infty$, we discard the remaining terms. From the resulting equation we recover a Gaussian equilibrium distribution for $\xi$, or equivalently for $\sqrt{t}(w - w^*) \equiv \sqrt{t}v$ where $v$ is called the *weight error*. The asymptotically normal distribution for $\sqrt{t}v$ has variance $\sigma^2_{\sqrt{t}v}$ from which the asymptotic expected squared weight error can be derived

$$\lim_{t \to \infty} E[|v|^2] = \sigma^2_{\sqrt{t}v} \frac{1}{t} = \frac{\eta_0^2 \alpha_2^{(0)}}{2\eta_0 G^* - 1} \frac{1}{t} \quad (7)$$

where $G^* \equiv G(w^*)$ is the curvature at the local optimum.

Positive $\sigma_{\sqrt{t}v}$ requires $\eta_0 > 1/(2G^*)$. If this condition is *not* met the expected squared weight offset converges as $(1/t)^{1-2\eta_0 G^*}$, *slower* than $1/t$ [5, for example, and references therein]. The above confirms the classical results [1] on asymptotic normality and convergence rate for $1/t$ annealing.

For the case $0 < p < 1$, the second and third terms on the right hand side of (6) will dominate as $t \to \infty$. Again, we have a Gaussian equilibrium density for $\xi$. Consequently $\sqrt{t^p}v$ is asymptotically normal with variance $\sigma^2_{\sqrt{t^p}v}$ leading to the expected squared weight error

$$E[|v|^2] = \sigma^2_{\sqrt{t^p}v} \frac{1}{t^p} = \frac{\eta_0 \alpha_2^{(0)}}{2G} \frac{1}{t^p} \quad (8)$$

Notice that the convergence is *slower* than $1/t$ and that there is *no* critical value of the learning rate to obtain a sensible equilibrium distribution. (See [9] for earlier results on $1/t^p$ annealing.)

The generalization error follows the same decay rate as the expected weight offset. In one dimension, the expected squared weight offset is directly related to excess generalization error (the generalization error minus the least generalization error achievable) $\epsilon_g = G E[v^2]$. In multiple dimensions, the expected squared weight offset, together with the maximum and minimum eigenvalues of $G^*$ provide upper and lower bounds on the excess generalization error proportional to $E[|v|^2]$, with the criticality condition on $G^*$ (for $p = 1$) replaced with an analogous condition on its eigenvalues.

## 3   Order parameters

In the Master equation approach, one focuses attention on the weight space distribution $P(w, t)$ and calculates quantities of interested by averaging over this density. An alternative approach is to choose a smaller set of *macroscopic* variables that are sufficient for describing principal properties of the system such as the generalization error (in contrast to the evolution of the weights $w$ which are *microscopic*).

Formally, one can replace the parameter dynamics presented in Eq.(1) by the corresponding equation for macroscopic observables which can be easily derived from the corresponding expressions for $w$. By choosing an appropriate set of macroscopic variables and invoking the thermodynamic limit (i.e., looking at systems where the number of parameters is infinite), one obtains point distributions for the order parameters, rendering the dynamics deterministic.

Several researchers [6, 7] have employed this approach for calculating the training dynamics of a soft committee machine (SCM) . The SCM maps inputs $x \in \Re^N$ to a scalar, through a model $\rho(\mathbf{w}, x) = \sum_{i=1}^{K} g(\mathbf{w}_i \cdot x)$. The activation function of the hidden units is $g(u) \equiv \mathrm{erf}(u/\sqrt{2})$ and $\mathbf{w}_i$ is the set of input-to-hidden adaptive weights for the $i = 1 \dots K$ hidden nodes. The hidden-to-output weights are set to 1. This architecture preserves most of the properties of the learning dynamics and the evolution of the generalization error as a general two-layer network, and the formalism can be easily extended to accommodate adaptive hidden-to-output weights [10].

Input vectors $x$ are independently drawn with zero mean and unit variance, and the corresponding targets $y$ are generated by deterministic teacher network corrupted by additive Gaussian output noise of zero mean and variance $\sigma_\nu^2$. The teacher network is also a SCM, with input-to-hidden weights $\mathbf{w}_i^*$. The order parameters sufficient to close the dynamics, and to describe the network generalization error are overlaps between various input-to-hidden vectors $\mathbf{w}_i \cdot \mathbf{w}_k \equiv Q_{ik}$, $\mathbf{w}_i \cdot \mathbf{w}_n^* \equiv R_{in}$, and $\mathbf{w}_n^* \cdot \mathbf{w}_m^* \equiv T_{nm}$ .

Network performance is measured in terms of the generalization error $\epsilon_g(\mathbf{w}) \equiv \langle 1/2\,[\,\rho(\mathbf{w}, x) - y\,]^2 \rangle_x$. The generalization error can be expressed in closed form in terms of the order parameters in the thermodynamic limit $(N \to \infty)$. The dynamics of the latter are also obtained in closed form [7]. These dynamics are coupled non-linear ordinary differential equations whose solution can only be obtained through numerical integration. However, the *asymptotic* behavior in the case of annealed learning *is* amenable to analysis, and this is one of the primary results of the paper.

We assume an isotropic teacher $T_{nm} = \delta_{nm}$ and use this symmetry to reduce the system to a vector of four order parameters $u^T = (r, q, s, c)$ related to the overlaps by $R_{in} = \delta_{in}(1+r) + (1-\delta_{in})s$ and $Q_{ik} = \delta_{ik}(1+q) + (1-\delta_{ik})c$.

With learning rate annealing and $\lim_{t \to \infty} u = 0$ we describe the dynamics in this vicinity by a linearization of the equations of motion in [7]. The linearization is

$$\frac{d}{dt}\mathbf{u} = \eta M \mathbf{u} + \eta^2 \sigma_\nu^2 \mathbf{b}\,, \qquad (9)$$

where $\sigma_\nu^2$ is the noise variance, $\mathbf{b}^T = \frac{2}{\pi}\left(0, 1/\sqrt{3}, 0, 1/2\right)$, $\eta = \eta_0/t^p$, and $M$ is

$$M = \frac{2}{3\sqrt{3}\pi}\begin{pmatrix} -4 & \frac{3}{2} & -\frac{9}{4}(K-1)\sqrt{(3)} & \frac{3}{4}(K-1)\sqrt{3} \\ 4 & -3 & \frac{3}{2}(K-1)\sqrt{3} & -\frac{3}{2}(K-1)\sqrt{3} \\ -\frac{3}{2}\sqrt{3} & \frac{3}{8}\sqrt{3} & -\frac{3}{2}(K-2)+\frac{2}{\sqrt{3}} & 0 \\ 3\sqrt{3} & -\frac{9}{4}\sqrt{3} & 3\sqrt{3}(K-2)+\frac{2}{\sqrt{3}} & -3\sqrt{3}(K-2)+\frac{2}{\sqrt{3}} \end{pmatrix}, \qquad (10)$$

The asymptotic equations of motion (9) were derived by dropping terms of order $\mathcal{O}(\eta\|\mathbf{u}\|^2)$ and higher, *and* terms of order $\mathcal{O}(\eta^2\,\mathbf{u})$. While the latter are linear in the order parameters, they are dominated by the $\eta\,\mathbf{u}$ and $\eta^2\sigma_\nu^2\mathbf{b}$ terms in (9) as $t \to \infty$.

This choice of truncations sheds light on the approach to equilibrium that is not implicit in the master equation approach. In the latter, the dominant terms for the asymptotics of (6) were identified by time scale of the coefficients, there was no identification of system observables that signal when the asymptotic regime is entered. For the order parameter approach, the conditions for validity of the asymptotic approximations are cast in terms of system observables $\eta \mathbf{u}$ vs $\eta^2 \mathbf{u}$ vs $\eta^2 \sigma_\nu^2$.

The solution to (9) is

$$\mathbf{u}(t) = \gamma(t, t_0) \, \mathbf{u}_0 + \sigma_\nu^2 \, \beta(t, t_0) \, \mathbf{b} \tag{11}$$

where $\mathbf{u}_0 \equiv \mathbf{u}(t_0)$ and

$$\gamma(t, t_0) = \exp \left\{ M \int_{t_0}^{t} d\tau \, \eta(\tau) \right\} \quad \text{and} \quad \beta(t, t_0) = \int_{t_0}^{t} d\tau \, \gamma(t, \tau) \, \eta^2(\tau) . \tag{12}$$

The asymptotic order parameter dynamics allow us to compute the generalization error (to first order in $\mathbf{u}$)

$$\epsilon_l = \frac{K}{\pi} \left( \frac{1}{\sqrt{3}} (q - 2r) + \frac{K-1}{2} (c - 2s) \right) . \tag{13}$$

Using the solution of Eq.(11), the generalization error consists of two pieces: a contribution depending on the actual initial conditions $\mathbf{u}_0$ and a contribution due to the second term on the r.h.s. of Eq.(11), independent of $\mathbf{u}_0$. The former decays more rapidly than the latter, and we ignore it in what follows. Asymptotically, the generalization error is of the form $\epsilon_l = \sigma_\nu^2 (c_1 \theta_1(t) + c_2 \theta_2(t))$, where $c_i$ are $K$ dependent coefficients, and $\theta_i$ are eigenmodes that evolve as

$$\theta_i = -\frac{\eta_0^2}{1 + \alpha_i \eta_0} \left[ \frac{1}{t} - t^{\alpha_i \eta_0} t_0^{-(\alpha_i \eta_0 + 1)} \right] . \tag{14}$$

with eigenvalues (Fig. 1(a))

$$\alpha_1 = -\frac{1}{\pi} \left( \frac{4}{\sqrt{3}} - 2 \right) \quad \text{and} \quad \alpha_2 = -\frac{1}{\pi} \left( \frac{4}{\sqrt{3}} + 2(K-1) \right) \tag{15}$$

The critical learning rate $\eta_0^{\text{crit}}$, above which the generalization decays as $1/t$ is, for $K \geq 2$,

$$\eta_0^{\text{crit}} = \max \left( -\frac{1}{\alpha_1}, -\frac{1}{\alpha_2} \right) = \frac{\pi}{4/\sqrt{3} - 2} . \tag{16}$$

For $\eta_0 > \eta_0^{\text{crit}}$ both modes $\theta_i, i = 1, 2$ decay as $1/t$, and so

$$\epsilon_l = -\sigma_\nu^2 \eta_0^2 \left( \frac{c_1}{1 + \alpha_1 \eta_0} + \frac{c_2}{1 + \alpha_2 \eta_0} \right) \frac{1}{t} \equiv \sigma_\nu^2 f(\eta_0, K) \frac{1}{t} . \tag{17}$$

Minimizing the prefactor $f(\eta_0, K)$ in (17) minimizes the asymptotic error. The values $\eta_0^{\text{opt}}(K)$ are shown in Fig. 1(b), where the special case of $K = 1$ (see below) is also included: There is a significant difference between the values for $K = 1$ and $K = 2$ and a rather weak dependence on $K$ for $K \geq 2$. The sensitivity of the generalization error decay factor on the choice of $\eta_0$ is shown in Fig. 1(c).

The influence of the noise strength on the generalization error can be seen directly from (17): the noise variance $\sigma_\nu^2$ is just a prefactor scaling the $1/t$ decay. Neither the value for the critical nor for the optimal $\eta_0$ is influenced by it.

The calculation above holds for the case $K = 1$ (where $c$ and $s$ and the mode $\theta_1$ are absent). In this case

$$\eta_0^{\text{opt}}(K = 1) = 2\eta_0^{\text{crit}}(K = 1) = -\frac{2}{\alpha_2} = \frac{\sqrt{3}\pi}{2}. \tag{18}$$

Finally, for the general annealing schedule of the form $\eta = \eta_0/t^p$ with $0 < p < 1$ the equations of motion (11) can be investigated, and one again finds $1/t^p$ decay.

## 4   Discussion and summary

We employed master equation and order parameter approaches to study the convergence of on-line learning under different annealing schedules. For the $1/t$ annealing schedule, the small noise expansion provides a critical value of $\eta_0$ (7) in terms of the curvature, above which $\sqrt{t}\,v$ is asymptotically normal, and the generalization decays as $1/t$. The approach is general, but requires knowledge of the first two jump moments in the asymptotic regime for calculating the relevant properties.

By restricting the order parameters approach to a symmetric task characterized by a set of isotropic teacher vectors, one can explicitly solve the dynamics in the asymptotic regime for any number of hidden nodes, and provide explicit expressions for the decaying generalization error and for the critical (16) and optimal learning rate prefactors for any number of hidden nodes $K$. Moreover, one can study the sensitivity of the generalization error decay to the choice of this prefactor. Similar results have been obtained for the critical learning rate prefactors using both methods, and both methods have been used to study general $1/t^p$ annealing. However, the order parameters approach enables one to gain a complete description of the dynamics and additional insight by restricting the task examined. Finally the order parameters approach expresses the dynamics in terms of ordinary differential equations, rather than partial differential equations; a clear advantage for numerical investigations.

The order parameter approach provides a potentially helpful insight on the passage into the asymptotic regime. Unlike the truncation of the small noise expansion, the truncation of the order parameter equations to obtain the asymptotic dynamics is couched in terms of system observables (c.f. the discussion following (10)). That is, one knows exactly which observables must be dominant for the system to be in the asymptotic regime. Equivalently, starting from the full equations, the order parameters approach can tell us when the system is close to the equilibrium distribution.

Although we obtained a full description of the asymptotic dynamics, it is still unclear how relevant it is in the larger picture which includes *all* stages of the training process, as in many cases it takes a prohibitively long time for the system to reach the asymptotic regime. It would be interesting to find a way of extending this framework to gain insight into earlier stages of the learning process.

**Acknowledgements:** DS and BS would like to thank the Leverhulme Trust for their support (F/250/K). TL thanks the International Human Frontier Science Program (SF 473-96), and the NSF (ECS-9704094) for their support.

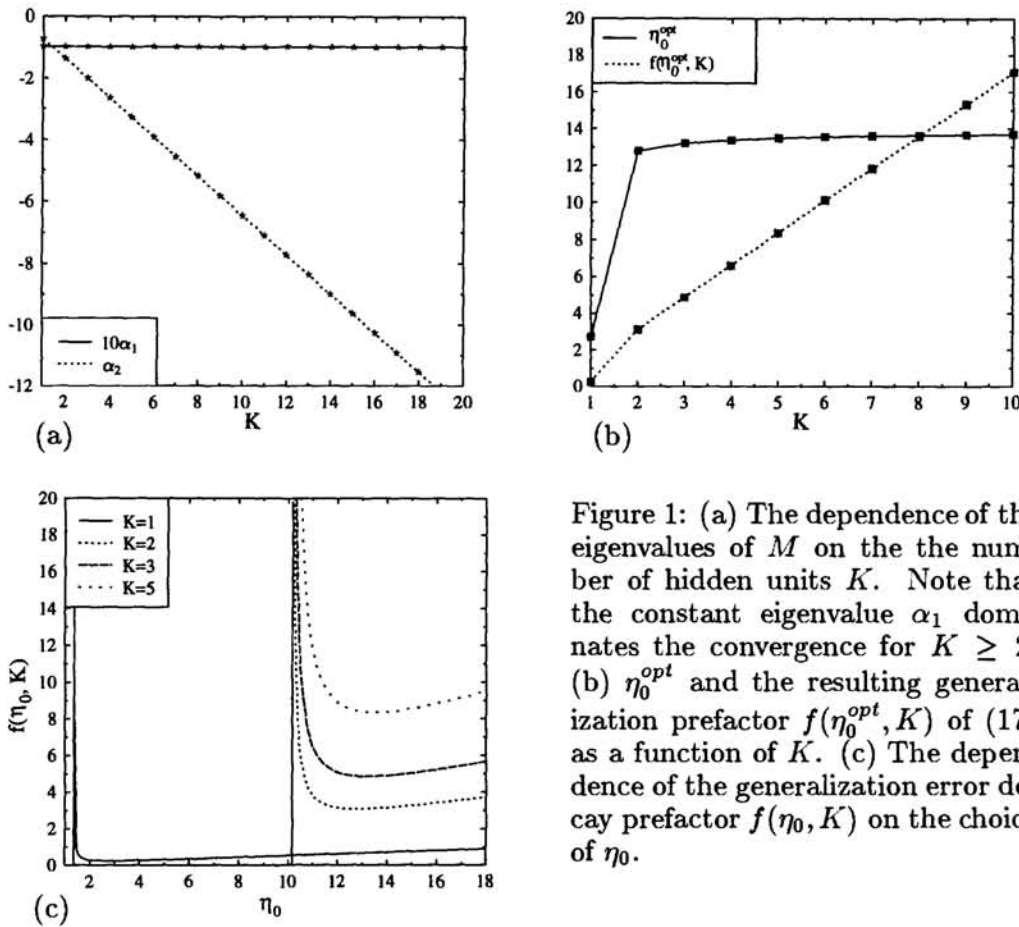

Figure 1: (a) The dependence of the eigenvalues of $M$ on the the number of hidden units $K$. Note that the constant eigenvalue $\alpha_1$ dominates the convergence for $K \geq 2$. (b) $\eta_0^{opt}$ and the resulting generalization prefactor $f(\eta_0^{opt}, K)$ of (17) as a function of $K$. (c) The dependence of the generalization error decay prefactor $f(\eta_0, K)$ on the choice of $\eta_0$.

## References

[1] V. Fabian. *Ann. Math. Statist.*, **39**, 1327 1968.

[2] L. Goldstein. Technical Report DRB-306, Dept. of Mathematics, University of Southern California, LA, 1987.

[3] T. M. Heskes and B. Kappen, *Phys. Rev. A* **44**, 2718 (1991).

[4] T. K. Leen and J. E. Moody. In Giles, Hanson, and Cowan, editors, *Advances in Neural Information Processing Systems*, 5, 451, San Mateo, CA, 1993. Morgan Kaufmann.

[5] T. K. Leen and G. B. Orr. In J.D. Cowan, G. Tesauro, and J. Alspector, editors, *Advances in Neural Information Processing Systems 6*, 477 ,San Francisco, CA., 1994. Morgan Kaufmann Publishers.

[6] M. Biehl and H. Schwarze, *J. Phys. A* **28**, 643 (1995).

[7] D. Saad and S.A. Solla *Phys. Rev. Lett.* **74**, 4337 (1995) and *Phys. Rev. E* **52** 4225 (1995).

[8] G. B. Orr. *Dynamics and Algorithms for Stochastic Search*. PhD thesis, Oregon Graduate Institute, October 1996.

[9] Naama Barkai. *Statistical Mechanics of Learning*. PhD thesis, Hebrew University of Jerusalem, August 1995.

[10] P. Riegler and M. Biehl *J. Phys. A* **28**, L507 (1995).
